# Efficient Simulation of Biological Neural Networks on Massively Parallel Supercomputers with Hypercube Architecture

**Ernst Niebur**
Computation and Neural Systems
California Institute of Technology
Pasadena, CA 91125, USA

**Dean Brettle**
Booz, Allen and Hamilton, Inc.
8283 Greensboro Drive
McLean, VA 22102-3838, USA

## Abstract

We present a neural network simulation which we implemented on the massively parallel Connection Machine 2. In contrast to previous work, this simulator is based on biologically realistic neurons with nontrivial single-cell dynamics, high connectivity with a structure modelled in agreement with biological data, and preservation of the temporal dynamics of spike interactions. We simulate neural networks of 16,384 neurons coupled by about 1000 synapses per neuron, and estimate the performance for much larger systems. Communication between neurons is identified as the computationally most demanding task and we present a novel method to overcome this bottleneck. The simulator has already been used to study the primary visual system of the cat.

## 1 INTRODUCTION

Neural networks have been implemented previously on massively parallel supercomputers (Fujimoto et al., 1992, Zhang et al., 1990). However, these are implementations of *artificial*, highly simplified neural networks, while our aim was explicitly to provide a simulator for *biologically realistic* neural networks. There is also at least one implementation of biologically realistic neuronal systems on a moderately

parallel but powerful machine (De Schutter and Bower, 1992), but the complexity of the used neuron model makes simulation of larger numbers of neurons impractical. Our interest here is to provide an efficient simulator of large neural networks of cortex and related subcortical structures.

The most important characteristics of the neuronal systems we want to simulate are the following:

- Cells are highly interconnected (several thousand connections per cell) but far from fully interconnected.

- Connections do not follow simple deterministic rules (like, e.g., nearest neighbor connections).

- Cells communicate with each other via delayed spikes which are binary events ("all-or-nothing").

- Such communication events are short (1 $ms$) and infrequent (1 to 100 per second).

- The temporal fine structure of the spike trains may be an important information carrier (Kreiter and Singer, 1992, Richmond and Optican, 1990, Softky and Koch, 1993).

## 2   IMPLEMENTATION

The biological network was modelled as a set of improved integrate-and-fire neurons which communicate with each other via delayed impulses (spikes). The single-cell model and details of the connectivity have been described in refs. (Wehmeier et al., 1989, Wörgötter et al., 1991).

Despite the rare occurrence of action potentials, their processing accounts for the major workload of the machine. The efficient implementation of inter-neuron communication is therefore *the* decisive factor which determines the efficacy of the simulator implementation. By "spike propagation" we denote the process by which a neuron communicates the occurrence of an action potential to all its postsynaptic partners. While the most efficient computation of the neuronal equations is obtained by mapping each neuron on one processor, this is very inefficient for spike propagation. This is due to the fact that spikes are rare events and that in the SIMD architecture used, each processor has to wait for the completion of the current tasks of all other processors. Therefore, only very few processors are active at any given time step. A more efficient data representation than provided by this "direct" algorithm is shown in Fig. 1. In this "transposed" scheme, a processor changes its role from simulating one of the neurons to simulating one synapse, which is, in general, *not* a synapse of the neuron simulated by the processor (see legend of Fig. 1). At any given time step, the addresses of the processors representing spiking neurons are broadcast along binary trees which are implemented efficiently (in time complexity $log_2 M$ for $M$ processors) in a hypercube architecture such as the CM-2. We obtain further computational efficiency by dividing the processor array into "partitions" of size $M$ and by implementing partially parallel I/O scheduling (both not discussed here).

| 1 | 2 | 3 | 4 | 5 | | | i | | | | M-1 | M |
|---|---|---|---|---|---|---|---|---|---|---|---|---|
| | | | | | | | | | | | | |
| 1,1 | 1,2 | 1,3 | 1,4 | · | | · | 1,i | · | | | · | 1,M |
| 2,1 | 2,2 | 2,3 | 2,4 | · | | · | 2,i | · | | | · | 2,M |
| 3,1 | 3,2 | 3,3 | 3,4 | · | | · | 3,i | · | | | · | 3,M |
| i,1 | i,2 | i,3 | i,4 | · | | · | i,i | · | | | · | i,M |
| ⬇ | ⬇ | ⬇ | ⬇ | | | | ⬇ | | | | | ⬇ |
| M,1 | M,2 | M,3 | M,4 | · | | · | M,i | · | | | · | M,M |

Figure 1: Transposed storage method for connections. The storage space for each of the $N$ processors is represented by a vertical column. A small part of this space is used for the time-dependent variables describing each of the $N$ neurons (upper part of each column, "Cell data"). The main part of the storage is used for datasets consisting of the addresses, weights and delays of the synapses ("Synapse data"), represented by the indices $i, j$ in the figure. For instance, "1, 1" stands for the first synapse of neuron 1, "1, 2" for the second synapse of this neuron and so on. Note that the storage space of processor $i$ does not hold the synapses of neuron $i$. If neuron $i$ generates a spike, all $M$ processors are used for propagating the spike (black arrows)

# 3   PERFORMANCE ANALYSIS

In order to accurately compare the performance of the described spike propagation algorithms, we implemented both the direct algorithm and the transposed algorithm and compared their performances with analytical estimates.

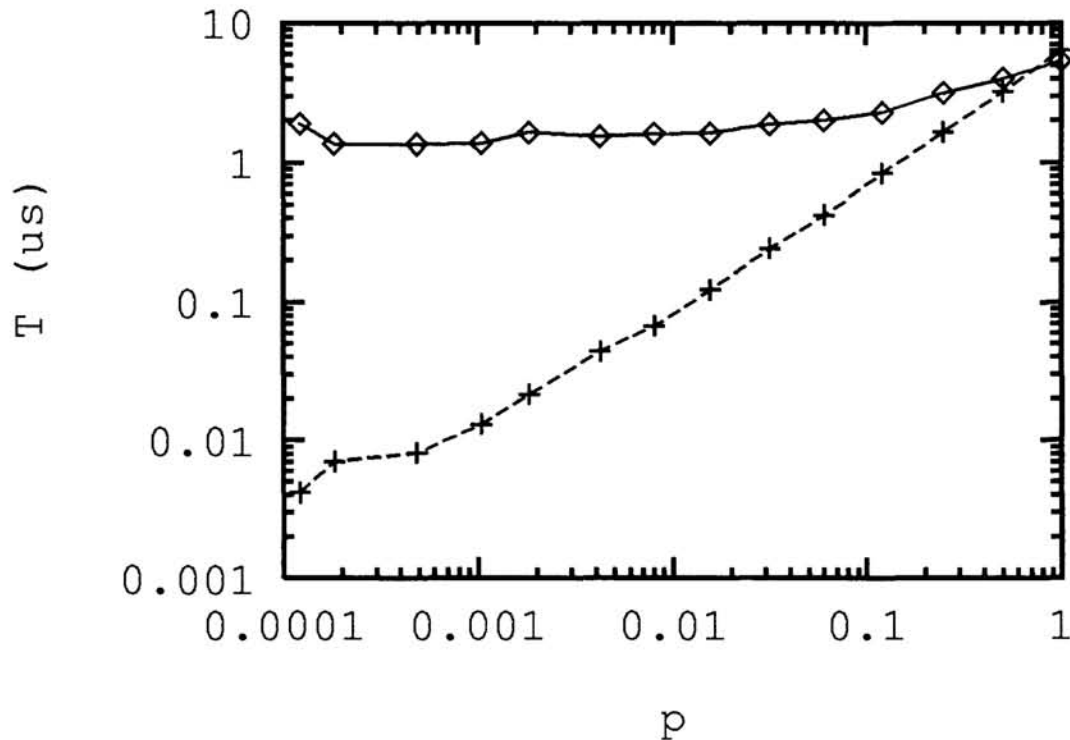

Figure 2: Execution time for the direct algorithm (diamonds) and the transposed algorithm (crosses) as function of the spiking probability $p$ for each cell. If all cells fire at each time step, there is no advantage for the transposed algorithm; in fact, it is at a disadvantage due to the overhead discussed in the text. Therefore, the two curves cross at a value just below $p = 1$. As expected, the largest difference between them is found for the smallest values of $p$.

Figure 2 compares the time required for the direct algorithm to the time required for the transposed algorithm as a function of $p$, the average number of spikes per neuron per time step. Note that while the time required rises much more rapidly for the transposed algorithm than the direct algorithm, it takes significantly less time for $p < 0.5$. The peak speedup was a factor of 454 which occurred at $p = 0.00012$ (or 1.2 impulses per second at a timestep of 0.1ms, corresponding approximately to spontaneous spike rates). The absolutely highest possible speedup, obtained if there is exactly one spike in every partition at every time step, is equal to $M$ ($M = 1024$ in this simulation). The average speedup is determined by the maximal number of spiking neurons per time step in *any* partition, since the processors in all partitions have to wait until the last partition has propagated all of its spikes. The average maximal number of spikes in a system of $N$ partitions, each one consisting of $M$

neurons is

$$\overline{N_{max}}(p, M, N) = \sum_{k=0}^{M} k \sum_{m=1}^{N} \binom{N}{m} \Pi(k)^m \hat{\Pi}(k)^{N-m} \qquad (1)$$

where $p$ is the spiking probability of one cell, $\Pi(k)$ is the probability that a given partition has $k$ spikes and

$$\hat{\Pi}(k) = \sum_{i=0}^{k-1} \Pi(i) \qquad (2)$$

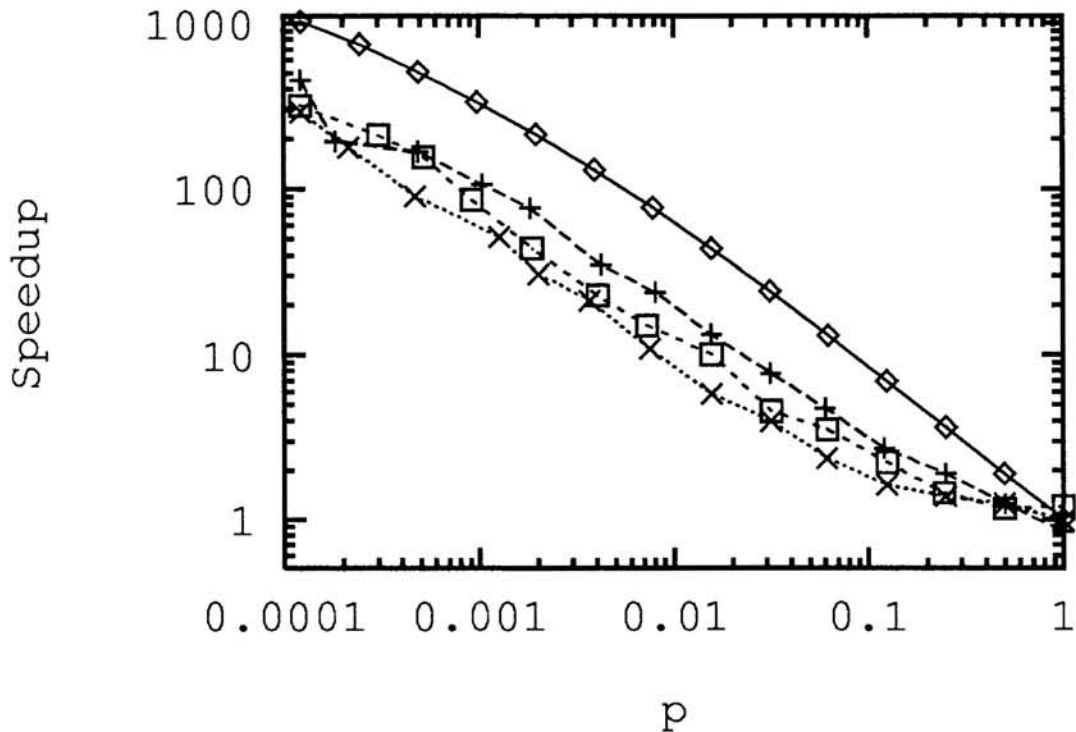

Figure 3: Speedup of the transposed algorithm over the direct algorithm as a function of $p$ for different VP ratios; $M = 1024$. The ideal speedup (uppermost curve; diamonds), computed in eq. 3 essentially determines the observed speedup. (lower curves; "+" signs: VP-ratio=1, diamonds: VP-raio=2, crosses: VP-ratio=4.). The difference between the ideal and the effectively obtained speedup is due to communication and other overhead of the transposed algorithm. Note that the difference in speedup for different VP ratios (difference between lower curves) is relatively small, which shows that the penalty for using larger neuron numbers is not large. As expected, the speedup approaches unity for $p \approx 1$ in all cases.

It can be shown that for independent neurons and for low spike rates, $\Pi(k)$ is the Poisson distribution and $\hat{\Pi}(k)$ the incomplete $\Gamma$ function. The average maximal

number of spikes for $M = 1024$ and different values of $p$ (eq. 1) can be shown to be a mildly growing function of the number of partitions which shows that the performance will not be limited crucially by changing the number of partitions. Therefore, the algorithm scales well with increasing network size and the performance-limiting factor is the activity level in the network and not the size of the network. This is also evident in Fig. 3 which shows the effectively obtained speedup compared to the ideal speedup, which would be obtained if the transposed algorithm were limited only by eq. 1 and would not require any additional communication or other overhead. Using $\overline{N_{max}}(p, M, N)$ from eq. 1 it is clear that this ideal speedup is given by

$$\frac{M}{\overline{N_{max}}(p, M, N)} \tag{3}$$

The difference between theory and experiment can be attributed to the time required for the spread operation and other additional overhead associated with the transposed algorithm. At $P = 0.0010$ (or 10 ips) the obtained speedup is a factor of 106.

## 4   VERY LARGE SYSTEMS

Using the full local memory of the machine and the "Virtual Processor" capability of the CM-2, the maximal number of neurons that can be simulated *without any change of algorithm* is as high as 4,194,304 ("4M"). Figure 3 shows that the speedup is reduced only slightly as the number of neurons increases, when the additional neurons are simulated by virtual processors. The performance is essentially limited by the mean network activity, whose effect is expressed by eq. 3, and the additional overhead originating from the higher "VP ratio" is small. This corroborates our earlier conclusion that the algorithm scales well with the size of the simulated system. Although we did not study the scaling of execution time with the size of the simulated system for more than 16,384 real processors, we expect the total execution time to be basically independent of the number of neurons, as long as additional neurons are distributed on additional processors.

**Acknowlegdements**

We thank U. Wehmeier and F. Wörgötter who provided us with the code for generating the connections, and G. Holt for his retina simulator. Discussions with C. Koch and F. Wörgötter were very helpful. We would like to thank C. Koch for his continuing support and for providing a stimulating research atmosphere. We also acknowledge the Advanced Computing Laboratory of Los Alamos National Laboratory, Los Alamos, NM 87545. Some of the numerical work was performed on computing resources located at this facility. This work was supported by the National Science Foundation, the Office of Naval Research, and the Air Force Office of Scientific Research.

# References

De Schutter E. and Bower J.M. (1992). Purkinje cell simulation on the Intel Touchstone Delta with GENESIS. In Mihaly T. and Messina P., editors, *Proceedings of the Grand Challenge Computing Fair*, pages 268–279. CCSF Publications, Caltech, Pasadena CA.

Fujimoto Y., Fukuda N., and Akabane T. (1992). Massively parallel architectures for large scale neural network simulations. *IEEE Transactions on Neural Networks*, 3(6):876–888.

Kreiter A.K. and Singer W. (1992). Oscillatory neuronal responses in the visual-cortex of the awake macaque monkey. *Europ. J. Neurosci.*, 4(4):369–375.

Richmond B.J. and Optican L.M. (1990). Temporal encoding of two-dimensional patterns by single units in primate primary visual cortex. II: Information transmission. *J. Neurophysiol.*, 64:370–380.

Softky W. and Koch C. (1993). The highly irregular firing of cortical-cells is inconsistent with temporal integration of random epsps. *J. Neurosci.*, 13(1):334–350.

Wehmeier U., Dong D., Koch C., and van Essen D. (1989). Modeling the visual system. In Koch C. and Segev I., editors, *Methods in Neuronal Modeling*, pages 335–359. MIT Press, Cambridge, MA.

Wörgötter F., Niebur E., and Koch C. (1991). Isotropic connections generate functional asymmetrical behavior in visual cortical cells. *J. Neurophysiol.*, 66(2):444–459.

Zhang X., Mckenna M., Mesirov J., and Waltz D. (1990). An efficient implementation of the back-propagation algorithm on the Connection Machine CM-2. In Touretzky D.S., editor, *Neural Information Processing Systems 2*, pages 801–809. Morgan-Kaufmann, San Mateo, CA.
